# Learning vehicular dynamics, with application to modeling helicopters

**Pieter Abbeel**
Computer Science Dept.
Stanford University
Stanford, CA 94305

**Varun Ganapathi**
Computer Science Dept.
Stanford University
Stanford, CA 94305

**Andrew Y. Ng**
Computer Science Dept.
Stanford University
Stanford, CA 94305

## Abstract

We consider the problem of modeling a helicopter's dynamics based on state-action trajectories collected from it. The contribution of this paper is two-fold. First, we consider the linear models such as learned by CIFER (the industry standard in helicopter identification), and show that the linear parameterization makes certain properties of dynamical systems, such as inertia, fundamentally difficult to capture. We propose an alternative, acceleration based, parameterization that does not suffer from this deficiency, and that can be learned as efficiently from data. Second, a Markov decision process model of a helicopter's dynamics would explicitly model only the one-step transitions, but we are often interested in a model's predictive performance over longer timescales. In this paper, we present an efficient algorithm for (approximately) minimizing the prediction error over long time scales. We present empirical results on two different helicopters. Although this work was motivated by the problem of modeling helicopters, the ideas presented here are general, and can be applied to modeling large classes of vehicular dynamics.

## 1 Introduction

In the last few years, considerable progress has been made in finding good controllers for helicopters. [7, 9, 2, 4, 3, 8] In designing helicopter controllers, one typically begins by constructing a model for the helicopter's dynamics, and then uses that model to design a controller. In our experience, after constructing a simulator (model) of our helicopters, policy search [7] almost always learns to fly (hover) very well in simulation, but may perform less well on the real-life helicopter. These differences between simulation and real-life performance can therefore be directly attributed to errors in the simulator (model) of the helicopter, and building accurate helicopter models remains a key technical challenge in autonomous flight. Modeling dynamical systems (also referred to as system identification) is one of the most basic and important problems in control. With an emphasis on helicopter aerodynamics, in this paper we consider the problem of learning good dynamical models of vehicles.

Helicopter aerodynamics are, to date, somewhat poorly understood, and (unlike most fixed-wing aircraft) no textbook models will accurately predict the dynamics of a helicopter from only its dimensions and specifications. [5, 10] Thus, at least part of the dynamics must be learned from data. CIFER® (Comprehensive Identification from Frequency Responses) is the industry standard for learning helicopter (and other rotorcraft) models from data. [11, 6]

CIFER uses frequency response methods to identify a linear model.

The models obtained from CIFER fail to capture some important aspects of the helicopter dynamics, such as the effects of inertia. Consider a setting in which the helicopter is flying forward, and suddenly turns sideways. Due to inertia, the helicopter will continue to travel in the same direction as before, so that it has "sideslip," meaning that its orientation is not aligned with its direction of motion. This is a non-linear effect that depends both on velocity and angular rates. The linear CIFER model is unable to capture this. In fact, the models used in [2, 8, 6] all suffer from this problem. The core of the problem is that the naive body-coordinate representation used in all these settings makes it fundamentally difficult for the learning algorithm to capture certain properties of dynamical systems such as inertia and gravity. As such, one places a significantly heavier burden than is necessary on the learning algorithm.

In Section 4, we propose an alternative parameterization for modeling dynamical systems that does not suffer from this deficiency. Our approach can be viewed as a hybrid of physical knowledge and learning. Although helicopter dynamics are not fully understood, there are also many properties—such as the direction and magnitude of acceleration due to gravity; the effects of inertia; symmetry properties of the dynamical system; and so on—which apply to *all* dynamical systems, and which are well-understood. All of this can therefore be encoded as prior knowledge, and there is little need to demand that our learning algorithms learn them. It is not immediately obvious how such prior knowledge can be encoded into a complex learning algorithm, but we will describe an acceleration based parameterization in which this can be done.

Given any model class, we can choose the parameter learning criterion used to learn a model within the class. CIFER finds the parameters that minimize a frequency domain error criterion. Alternatively, we can minimize the squared one-step prediction error in the time domain. Forward simulation on a held-out test set is a standard way to assess model quality, and we use it to compare the linear models learned using CIFER to the same linear models learned by optimizing the one-step prediction error. As suggested in [1], one can also learn parameters so as to optimize a "lagged criterion" that directly measures simulation accuracy—i.e., predictive accuracy of the model over long time scales. However, the EM algorithm given in [1] is expensive when applied in a continuous state-space setting. In this paper, we present an efficient algorithm that approximately optimizes the lagged criterion. Our experiments show that the resulting model consistently outperforms the linear models trained using CIFER or using the one-step error criterion. Combining this with the acceleration based parameterization results in our best helicopter model.

## 2 Helicopter state, input and dynamics

The helicopter state $s$ comprises its position $(x, y, z)$, orientation (roll $\phi$, pitch $\theta$, yaw $\omega$), velocity $(\dot{x}, \dot{y}, \dot{z})$ and angular velocity $(\dot{\phi}, \dot{\theta}, \dot{\omega})$. The helicopter is controlled via a 4-dimensional action space:

1. $u_1$ and $u_2$: The longitudinal (front-back) and latitudinal (left-right) cyclic pitch controls cause the helicopter to pitch forward/backward or sideways, and can thereby also affect acceleration in the longitudinal and latitudinal directions.

2. $u_3$: The tail rotor collective pitch control affects tail rotor thrust, and can be used to yaw (turn) the helicopter.

3. $u_4$: The main rotor collective pitch control affects the pitch angle of the main rotor's blades, by rotating the blades around an axis that runs along the length of the blade. As the main rotor blades sweep through the air, the resulting amount of upward thrust (generally) increases with this pitch angle; thus this control affects the main rotor's thrust.

Following standard practice in system identification ([8, 6]), the original 12-dimensional helicopter state is reduced to an 8-dimensional state represented in body (or robot-centric) coordinates $s^b = (\phi, \theta, \dot{x}, \dot{y}, \dot{z}, \dot{\phi}, \dot{\theta}, \dot{\omega})$. Where there is risk of confusion, we will use superscript $s$ and $b$ to distinguish between spatial (world) coordinates and body coordinates. The body coordinate representation specifies the helicopter state using a coordinate frame in which the $x$, $y$, and $z$ axes are forwards, sideways, and down relative to the current orientation of the helicopter, instead of north, east and down. Thus, $\dot{x}^b$ is the forward velocity, whereas $\dot{x}^s$ is the velocity in the northern direction. ($\phi$ and $\theta$ are always expressed in world coordinates, because roll and pitch relative to the body coordinate frame is always zero.) By using a body coordinate representation, we encode into our model certain "symmetries" of helicopter flight, such as that the helicopter's dynamics are the same regardless of its absolute position $(x, y, z)$ and heading $\omega$ (assuming the absence of obstacles). Even in the reduced coordinate representation, only a subset of the state variables needs to be modeled explicitly using learning. Given a model that predicts only the angular velocities $(\dot{\phi}, \dot{\theta}, \dot{\omega})$, we can numerically integrate to obtain the orientation $(\phi, \theta, \omega)$.

We can integrate the reduced body coordinate states to obtain the complete world coordinate states. Integrating body-coordinate angular velocities to obtain world-coordinate angles is nonlinear, thus the model resulting from this process is necessarily nonlinear.

## 3   Linear model

The linear model we learn with CIFER has the following form:

$$\dot{\phi}^b_{t+1} - \dot{\phi}^b_t = \left(C_\phi \dot{\phi}^b_t + C_1 (u_1)_t + D_1\right) \Delta t, \quad \dot{x}^b_{t+1} - \dot{x}^b_t = \left(C_x \dot{x}^b_t - g\theta_t\right) \Delta t,$$

$$\dot{\theta}^b_{t+1} - \dot{\theta}^b_t = \left(C_\theta \dot{\theta}^b_t + C_2 (u_2)_t + D_2\right) \Delta t, \quad \dot{y}^b_{t+1} - \dot{y}^b_t = \left(C_y \dot{y}^b_t + g\phi_t + D_0\right) \Delta t,$$

$$\dot{\omega}^b_{t+1} - \dot{\omega}^b_t = \left(C_\omega \dot{\omega}^b_t + C_3 (u_3)_t + D_3\right) \Delta t, \quad \dot{z}^b_{t+1} - \dot{z}^b_t = \left(C_z \dot{z}^b_t + g + C_4 (u_4)_t + D_4\right) \Delta t,$$

$$\phi_{t+1} - \phi_t = \dot{\phi}^b_t \Delta t, \qquad\qquad\qquad \theta_{t+1} - \theta_t = \dot{\theta}^b_t \Delta t.$$

Here $g = 9.81 m/s^2$ is the acceleration due to gravity and $\Delta t$ is the time discretization, which is 0.1 seconds in our experiments. The free parameters in the model are $C_x, C_y, C_z, C_\phi, C_\theta, C_\omega$, which model damping, and $D_0, C_1, D_1, C_2, D_2, C_3, D_3, C_4, D_4$, which model the influence of the inputs on the states.[1] This parameterization was chosen using the "coherence" feature selection algorithm of CIFER. CIFER takes as input the state-action sequence $\{(\dot{x}^b_t, \dot{y}^b_t, \dot{z}^b_t, \dot{\phi}^b_t, \dot{\theta}^b_t, \dot{\omega}^b_t, \phi_t, \theta_t, u_t)\}_t$ and learns the free parameters using a frequency domain cost function. See [11] for details.

Frequency response methods (as used in CIFER) are not the only way to estimate the free parameters. Instead, we can minimize the average squared prediction error of next state given current state and action. Doing so only requires linear regression. In our experiments (see Section 6) we compare the simulation accuracy over several time-steps of the differently learned linear models. We also compare to learning by directly optimizing the simulation accuracy over several time-steps. The latter approach is presented in Section 5.

## 4   Acceleration prediction model

Due to inertia, if a forward-flying helicopter turns, it will have sideslip (i.e., the helicopter will not be aligned with its direction of motion). The linear model is unable to capture the sideslip effect, since this effect depends non-linearly on velocity and angular rates. In fact, the models used in [2, 8, 6] all suffer from this problem. More generally, these models do not capture conservation of momentum well. Although careful engineering of (many) additional non-linear features might fix individual effects such as, e.g., sideslip, it is unclear how to capture inertia compactly in the naive body-coordinate representation.

From physics, we have the following update equation for velocity in body-coordinates:

$$(\dot{x}, \dot{y}, \dot{z})_{t+1}^b = R\left((\dot{\phi}, \dot{\theta}, \dot{\omega})_t^b\right) * \left((\dot{x}, \dot{y}, \dot{z})_t^b + (\ddot{x}, \ddot{y}, \ddot{z})_t^b \Delta t\right). \qquad (1)$$

Here, $R\left((\dot{\phi}, \dot{\theta}, \dot{\omega})_t^b\right)$ is the rotation matrix that transforms from the body-coordinate frame at time $t$ to the body-coordinate frame at time $t+1$ (and is determined by the angular velocity $(\dot{\phi}, \dot{\theta}, \dot{\omega})_t^b$ at time $t$); and $(\ddot{x}, \ddot{y}, \ddot{z})_t^b$ denotes the acceleration vector in body-coordinates at time $t$. Forces and torques (and thus accelerations) are often a fairly simple function of inputs and state. This suggests that a model which learns to predict the accelerations, and then uses Eqn. (1) to obtain velocity over time, may perform well. Such a model would naturally capture inertia, by using the velocity update of Eqn. (1). In contrast, the models of Section 3 try to predict changes in body-coordinate velocity. *But the change in body-coordinate velocity does not correspond directly to physical accelerations, because the body-coordinate velocity at times $t$ and $t + 1$ are expressed in different coordinate frames.* Thus, $\dot{x}_{t+1}^b - \dot{x}_t^b$ is not the forward acceleration—because $\dot{x}_{t+1}^b$ and $\dot{x}_t^b$ are expressed in different coordinate frames. To capture inertia, these models therefore need to predict not only the physical accelerations, but also the non-linear influence of the angular rates through the rotation matrix. This makes for a difficult learning problem, and puts an unnecessary burden on the learning algorithm. Our discussion above has focused on linear velocity, but a similar argument also holds for angular velocity.

The previous discussion suggests that we learn to *predict physical accelerations* and then integrate the accelerations to obtain the state trajectories. To do this, we propose:

$$\ddot{\phi}_t^b = C_\phi \dot{\phi}_t + C_1(u_1)_t + D_1, \quad \ddot{x}_t^b = C_x \dot{x}_t^b + (g_x)_t^b,$$
$$\ddot{\theta}_t^b = C_\theta \dot{\theta}_t + C_2(u_2)_t + D_2, \quad \ddot{y}_t^b = C_y \dot{y}_t^b + (g_y)_t^b + D_0,$$
$$\ddot{\omega}_t^b = C_\omega \dot{\omega}_t + C_3(u_3)_t + D_3, \quad \ddot{z}_t^b = C_z \dot{z}_t^b + (g_z)_t^b + C_4(u_4)_t + D_4.$$

Here $(g_x)_t^b, (g_y)_t^b, (g_z)_t^b$ are the components of the gravity acceleration vector in each of the body-coordinate axes at time $t$; and $C., D.$ are the free parameters to be learned from data. The model predicts accelerations in the body-coordinate frame, and is therefore able to take advantage of the same invariants as discussed earlier, such as invariance of the dynamics to the helicopter's $(x, y, z)$ position and heading $(\omega)$. Further, it additionally captures the fact that the dynamics are invariant to roll $(\phi)$ and pitch $(\theta)$ once the (known) effects of gravity are subtracted out.

Frequency domain techniques cannot be used to learn the acceleration model above, because it is non-linear. Nevertheless, the parameters can be learned as easily as for the linear model in the time domain: Linear regression can be used to find the parameters that minimize the squared error of the one-step prediction in acceleration.[2]

## 5  The lagged error criterion

To evaluate the performance of a dynamical model, it is standard practice to run a simulation using the model for a certain duration, and then compare the simulated trajectory with the real state trajectory. To do well on this evaluation criterion, it is therefore important for the dynamical model to give not only accurate one-step predictions, but also predictions that are accurate at longer time-scales. Motivated by this, [1] suggested learning the model parameters by optimizing the following "lagged criterion":

$$\sum_{t=1}^{T-H} \sum_{h=1}^{H} \|\hat{s}_{t+h|t} - s_{t+h}\|_2^2. \qquad (2)$$

Here, $H$ is the time horizon of the simulation, and $\hat{s}_{t+h|t}$ is the estimate (from simulation) of the state at time $t + h$ given the state at time $t$.

Unfortunately the EM-algorithm given in [1] is prohibitively expensive in our continuous state-action space setting. We therefore present a simple and fast algorithm for (approximately) minimizing the lagged criterion. We begin by considering a linear model with update equation:

$$s_{t+1} - s_t = As_t + Bu_t, \tag{3}$$

where $A, B$ are the parameters of the model. Minimizing the one-step prediction error would correspond to finding the parameters that minimize the expected squared difference between the left and right sides of Eqn. (3).

By summing the update equations for two consecutive time steps, we get that, for simulation to be exact over two time steps, the following needs to hold:

$$s_{t+2} - s_t = As_t + Bu_t + A\hat{s}_{t+1|t} + Bu_{t+1}. \tag{4}$$

Minimizing the expected squared difference between the left and right sides of Eqn. (4) would correspond to minimizing the two-step prediction error. More generally, by summing up the update equations for $h$ consecutive timesteps and then minimizing the left and right sides' expected squared difference, we can minimize the $h$-step prediction error. Thus, it may seem that we can directly solve for the parameters that minimize the lagged criterion of Eqn. (2) by running least squares on the appropriate set of linear combinations of state update equations.

The difficulty with this procedure is that the intermediate states in the simulation—for example, $\hat{s}_{t+1|t}$ in Eqn. (4)—are also an implicit function of the parameters $A$ and $B$. This is because $\hat{s}_{t+1|t}$ represents the result of a one-step simulation from $s_t$ using our model. Taking into account the dependence of the intermediate states on the parameters makes the right side of Eqn. (4) non-linear in the parameters, and thus the optimization is non-convex. If, however, we make an approximation and neglect this dependence, then optimizing the objective can be done simply by solving a linear least squares problem.

This gives us the following algorithm. We will alternate between a simulation step that finds the necessary predicted intermediate states, and a least squares step that solves for the new parameters.

LEARN-LAGGED-LINEAR:

1. Use least squares to minimize the one-step squared prediction error criterion to obtain an initial model $A^{(0)}, B^{(0)}$. Set $i = 1$.

2. For all $t = 1, \ldots, T$, $h = 1, \ldots, H$, simulate in the current model to compute $\hat{s}_{t+h|t}$.

3. Solve the following least squares problem:
$$(\bar{A}, \bar{B}) = \arg\min_{A,B} \sum_{t=1}^{T-H} \sum_{h=1}^{H} \|(s_{t+h} - s_t) - (\sum_{\tau=0}^{h-1} A\hat{s}_{t+\tau|t} + Bu_{t+\tau})\|_2^2.$$

4. Set $A^{(i+1)} = (1 - \alpha)A^{(i)} + \alpha\bar{A}$, $B^{(i+1)} = (1 - \alpha)B^{(i)} + \alpha\bar{B}$.[3]

5. If $\|A^{(i+1)} - A^{(i)}\| + \|B^{(i+1)} - B^{(i)}\| \leq \epsilon$ exit. Otherwise go back to step 2.

Our helicopter acceleration prediction model is not of the simple form $s_{t+1} - s_t = As_t + Bu_t$ described above. However, a similar derivation still applies: The change in velocity over several time-steps corresponds to the sum of changes in velocity over several single time-steps. Thus by adding the one-step acceleration prediction equations as given in Section 4, we might expect to obtain equations corresponding to the acceleration over several time-steps. However, the acceleration equations at different time-steps are in different coordinate frames. Thus we first need to rotate the equations and then add them. In the algorithm described below, we rotate all accelerations into the world coordinate frame. The acceleration equations from Section 4 give us $(\ddot{x}, \ddot{y}, \ddot{z})_t^b = A_{\text{pos}}s_t + B_{\text{pos}}u_t$, and

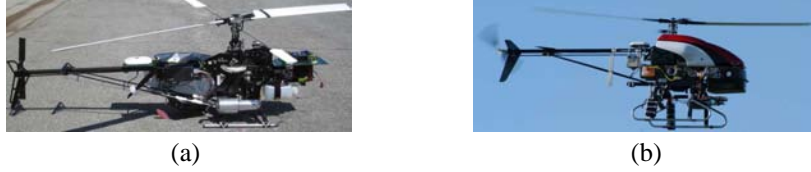

<div align="center">(a)                           (b)</div>

Figure 1: The XCell Tempest (a) and the Bergen Industrial Twin (b) used in our experiments.

$(\ddot{\phi}, \ddot{\theta}, \ddot{\omega})_t^b = A_{\mathrm{rot}} s_t + B_{\mathrm{rot}} u_t$, where $A_{\mathrm{pos}}, B_{\mathrm{pos}}, A_{\mathrm{rot}}, B_{\mathrm{rot}}$ are (sparse) matrices that contain the parameters to be learned.[4] This gives us the LEARN-LAGGED-ACCELERATION algorithm, which is identical to LEARN-LAGGED-LINEAR except that step 3 now solves the following least squares problems:

$$(\bar{A}_{\mathrm{pos}}, \bar{B}_{\mathrm{pos}}) = \arg\min_{A,B} \sum_{t=1}^{T-H} \sum_{h=1}^{H} \| \sum_{\tau=0}^{h-1} \hat{R}^{b_{t+\tau} \to s} \left( (\ddot{x}, \ddot{y}, \ddot{z})_{t+\tau}^b - (A\hat{s}_{t+\tau|t} + Bu_{t+\tau}) \right) \|_2^2$$

$$(\bar{A}_{\mathrm{rot}}, \bar{B}_{\mathrm{rot}}) = \arg\min_{A,B} \sum_{t=1}^{T-H} \sum_{h=1}^{H} \| \sum_{\tau=0}^{h-1} \hat{R}^{b_{t+\tau} \to s} \left( (\ddot{\phi}, \ddot{\theta}, \ddot{\omega})_{t+\tau}^b - (A\hat{s}_{t+\tau|t} + Bu_{t+\tau}) \right) \|_2^2$$

Here $\hat{R}^{b_t \to s}$ denotes the rotation matrix (estimated from simulation using the current model) from the body frame at time $t$ to the world frame.

## 6 Experiments

We performed experiments on two RC helicopters: an XCell Tempest and a Bergen Industrial Twin helicopter. (See Figure 1.) The XCell Tempest is a competition-class aerobatic helicopter (length 54", height 19"), is powered by a 0.91-size, two-stroke engine, and has an unloaded weight of 13 pounds. It carries two sensor units: a Novatel RT2 GPS receiver and a Microstrain 3DM-GX1 orientation sensor. The Microstrain package contains triaxial accelerometers, rate gyros, and magnetometers, which are used for inertial sensing. The larger Bergen Industrial Twin helicopter is powered by a twin cylinder 46cc, two-stroke engine, and has an unloaded weight of 18 lbs. It carries three sensor units: a Novatel RT2 GPS receiver, MicroStrain 3DM-G magnetometers, and an Inertial Science ISIS-IMU (triaxial accelerometers and rate gyros).

For each helicopter, we collected data from two separate flights. The XCell Tempest train and test flights were 800 and 540 seconds long, the Bergen Industrial Twin train and test flights were each 110 seconds long. A highly optimized Kalman filter integrates the sensor information and reports (at 100Hz) 12 numbers corresponding to the helicopter's state $(x, y, z, \dot{x}, \dot{y}, \dot{z}, \phi, \theta, \omega, \dot{\phi}, \dot{\theta}, \dot{\omega})$. The data is then downsampled to 10Hz before learning. For each of the helicopters, we learned the following models:

1. Linear-One-Step: The linear model from Section 3 trained using linear regression to minimize the one-step prediction error.

2. Linear-CIFER: The linear model from Section 3 trained using CIFER.

3. Linear-Lagged: The linear model from Section 3 trained minimizing the lagged criterion.

4. Acceleration-One-Step: The acceleration prediction model from Section 4 trained using linear regression to minimize the one step prediction error.

5. Acceleration-Lagged: The acceleration prediction model from Section 4 trained minimizing the lagged criterion.

For Linear-Lagged and Acceleration-Lagged we used a horizon $H$ of two seconds (20 simulation steps). The CPU times for training the different algorithms were: Less than one second for linear regression (algorithms 1 and 4 in the list above); one hour 20 minutes (XCell Tempest data) or 10 minutes (Bergen Industrial Twin data) for the lagged criteria (algorithms 3 and 5 above); about 5 minutes for CIFER. Our algorithm optimizing the lagged criterion appears to converge after at most 30 iterations. Since this algorithm is only approximate, we can then use coordinate descent search to further improve the lagged criterion.[5] This coordinate descent search took an additional four hours for the XCell Tempest data and an additional 30 minutes for the Bergen Industrial Twin data. We report results both with and without this coordinate descent search. Our results show that the algorithm presented in Section 5 works well for fast approximate optimization of the lagged criterion, but that locally greedy search (coordinate descent) may then improve it yet further.

For evaluation, the test data was split in consecutive non-overlapping two second windows. (This corresponds to 20 simulation steps, $s_0, \ldots, s_{20}$.) The models are used to predict the state sequence over the two second window, when started in the true state $s_0$. We report the average squared prediction error (difference between the simulated and true state) at each timestep $t = 1, \ldots, 20$ throughout the two second window. The orientation error is measured by the squared magnitude of the minimal rotation needed to align the simulated orientation with the true orientation. Velocity, position, angular rate and orientation errors are measured in m/s, m, rad/s and rad (squared) respectively. (See Figure 2.)

We see that Linear-Lagged consistently outperforms Linear-CIFER and Linear-One-Step. Similarly, for the acceleration prediction models, we have that Acceleration-Lagged consistently outperforms Acceleration-One-Step. These experiments support the case for training with the lagged criterion.

The best acceleration prediction model, Acceleration-Lagged, is significantly more accurate than any of the linear models presented in Section 3. This effect is mostly present in the XCell Tempest data, which contained data collected from many different parts of the state space (e.g., flying in a circle); in contrast, the Bergen Industrial Twin data was collected mostly near hovering (and thus the linearization assumptions were somewhat less poor there).

## 7   Summary

We presented an acceleration based parameterization for learning vehicular dynamics. The model predicts accelerations, and then integrates to obtain state trajectories. We also described an efficient algorithm for approximately minimizing the lagged criterion, which measures the predictive accuracy of the algorithm over both short and long time-scales. In our experiments, learning with the acceleration parameterization and using the lagged criterion gave significantly more accurate models than previous approaches. Using this approach, we have recently also succeeded in learning a model for, and then autonomously flying, a "funnel" aerobatic maneuver, in which the helicopter flies in a circle, keeping the tail pointed at the center of rotation, and the body of the helicopter pitched backwards at a steep angle (so that the body of the helicopter traces out the surface of a funnel). (Details will be presented in a forthcoming paper.)

**Acknowledgments.** We give warm thanks to Adam Coates and to helicopter pilot Ben Tse for their help on this work.

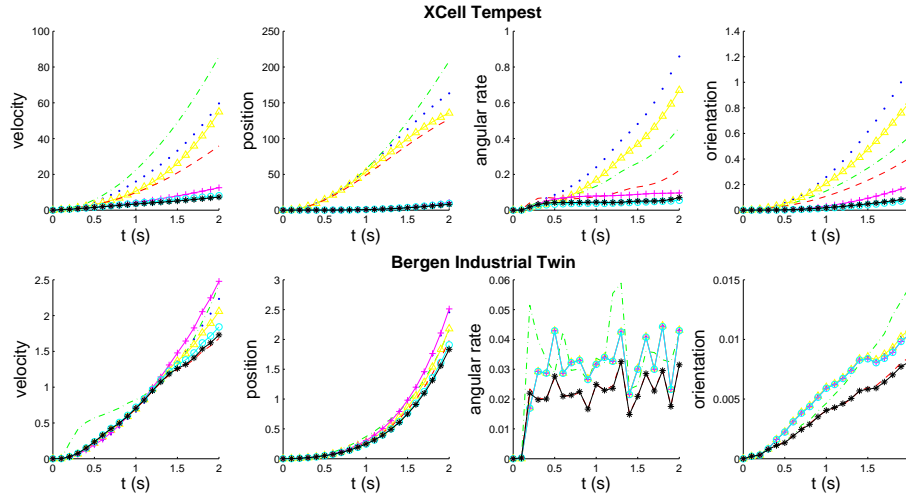

Figure 2: (Best viewed in color.) Average squared prediction errors throughout two-second simulations. Blue, dotted: Linear-One-Step. Green, dash-dotted: Linear-CIFER. Yellow, triangle: Linear-Lagged learned with fast, approximate algorithm from Section 5. Red, dashed: Linear-Lagged learned with fast, approximate algorithm from Section 5 followed by greedy coordinate descent search. Magenta, solid: Acceleration-One-Step. Cyan, circle: Acceleration-Lagged learned with fast, approximate algorithm from Section 5. Black,*: Acceleration-Lagged learned with fast, approximate algorithm from Section 5 followed by greedy coordinate descent search. The magenta, cyan and black lines (visually) coincide in the XCell position plots. The blue, yellow, magenta and cyan lines (visually) coincide in the Bergen angular rate and orientation plots. The red and black lines (visually) coincide in the Bergen angular rate plot. See text for details.

## Footnotes

[1] $D_0$ captures the sideways acceleration caused by the tail rotor's thrust.

[2] Note that, as discussed previously, the one-step difference of body coordinate velocities is not the acceleration. To obtain actual accelerations, the velocity at time $t + 1$ must be rotated into the body-frame at $t$ before taking the difference.

[3]This step of the algorithm uses a simple line search to choose the stepsize $\alpha$.

[4]For simplicity of notation we omit the intercept parameters here, but they are easily incorporated, e.g., by having one additional input which is always equal to one.

[5]We used coordinate descent on the criterion of Eqn. (2), but reweighted the errors on velocity, angular velocity, position and orientation to scale them to roughly the same order of magnitude.

## References

[1] P. Abbeel and A. Y. Ng. Learning first order Markov models for control. In *NIPS 18*, 2005.

[2] J. Bagnell and J. Schneider. Autonomous helicopter control using reinforcement learning policy search methods. In *International Conference on Robotics and Automation*. IEEE, 2001.

[3] V. Gavrilets, I. Martinos, B. Mettler, and E. Feron. Control logic for automated aerobatic flight of miniature helicopter. In *AIAA Guidance, Navigation and Control Conference*, 2002.

[4] V. Gavrilets, I. Martinos, B. Mettler, and E. Feron. Flight test and simulation results for an autonomous aerobatic helicopter. In *AIAA/IEEE Digital Avionics Systems Conference*, 2002.

[5] J. Leishman. *Principles of Helicopter Aerodynamics*. Cambridge University Press, 2000.

[6] B. Mettler, M. Tischler, and T. Kanade. System identification of small-size unmanned helicopter dynamics. In *American Helicopter Society, 55th Forum*, 1999.

[7] Andrew Y. Ng, Adam Coates, Mark Diel, Varun Ganapathi, Jamie Schulte, Ben Tse, Eric Berger, and Eric Liang. Autonomous inverted helicopter flight via reinforcement learning. In *International Symposium on Experimental Robotics*, 2004.

[8] Andrew Y. Ng, H. Jin Kim, Michael Jordan, and Shankar Sastry. Autnonomous helicopter flight via reinforcement learning. In *NIPS 16*, 2004.

[9] Jonathan M. Roberts, Peter I. Corke, and Gregg Buskey. Low-cost flight control system for a small autonomous helicopter. In *IEEE Int'l Conf. on Robotics and Automation*, 2003.

[10] J. Seddon. *Basic Helicopter Aerodynamics*. AIAA Education Series. America Institute of Aeronautics and Astronautics, 1990.

[11] M.B. Tischler and M.G. Cauffman. Frequency response method for rotorcraft system identification: Flight application to BO-105 couple rotor/fuselage dynamics. *Journal of the American Helicopter Society*, 1992.
